# SYNCHRONIZATION IN NEURAL NETS

## Jacques J. Vidal

University of California Los Angeles, Los Angeles, Ca. 90024

## John Haggerty*

## ABSTRACT

The paper presents an artificial neural network concept (the Synchronizable Oscillator Networks) where the instants of individual firings in the form of point processes constitute the only form of information transmitted between joining neurons. This type of communication contrasts with that which is assumed in most other models which typically are continuous or discrete value-passing networks. Limiting the messages received by each processing unit to time markers that signal the firing of other units presents significant implementation advantages.

In our model, neurons fire spontaneously and regularly in the absence of perturbation. When interaction is present, the scheduled firings are advanced or delayed by the firing of neighboring neurons. Networks of such neurons become global oscillators which exhibit multiple synchronizing attractors. From arbitrary initial states, energy minimization learning procedures can make the network converge to oscillatory modes that satisfy multi-dimensional constraints Such networks can directly represent routing and scheduling problems that consist of ordering sequences of events.

## INTRODUCTION

Most neural network models derive from variants of Rosenblatt's original perceptron and as such are value-passing networks. This is the case in particular with the networks proposed by Fukushima[1], Hopfield[2], Rumelhart[3] and many others. In every case, the inputs to the processing elements are either binary or continuous amplitude signals which are weighted by synaptic gains and subsequently summed (integrated). The resulting activation is then passed through a sigmoid or threshold filter and again produce a continuous or quantized output which may become the input to other neurons. The behavior of these models can be related to that of living neurons even if they fall considerably short of accounting for their complexity. Indeed, it can be observed with many real neurons that action potentials (spikes) are fired and propagate down the axonal branches when the internal activation reaches some threshold and that higher

John Haggerty is with Interactive Systems Los angeles
3030 W. 6th St. LA, Ca. 90020

input rates levels result in more rapid firing. Behind these traditional models, there is the assumption that the average frequency of action potentials is the carrier of information between neurons. Because of integration, the firings of individual neurons are considered effective only to the extent to which they contribute to the average intensities It is therefore assumed that the activity is simply "frequency coded". The exact timing of individual firing is ignored.

This view however does not cover some other well known aspects of neural communication. Indeed, the precise timing of spike arrivals can make a crucial difference to the outcome of some neural interactions. One classic example is that of pre-synaptic inhibition, a widespread mechanism in the brain machinery. Several studies have also demonstrated the occurrence and functional importance of precise timing or phase relationship between cooperating neurons in local networks[4, 5].

The model presented in this paper contrasts with the ones just mentioned in that in the networks each firing is considered as an individual output event. On the input side of each node, the firing of other nodes (the presynaptic neurons) either delay (inhibit) or advance (excite) the node firing. As seen earlier, this type of neuronal interaction which would be called phase-modulation in engineering systems, can also find its rationale in experimental neurophysiology. Neurophysiological plausibility however is not the major concern here. Rather, we propose to explore a potentially useful mechanism for parallel distributed computing. The merit of this approach for artificial neural networks is that digital pulses are used for internode communication instead of analog voltages. The model is particularly well suited to the time-ordering and sequencing found in a large class of routing and trajectory control problems.

### NEURONS AS SYNCHRONIZABLE OSCILLATORS:

In our model, the processing elements (the "neurons") are relaxation oscillators with built-in self-inhibition. A relaxation oscillator is a dynamic system that is capable of accumulating potential energy until some threshold or breakdown point is reached. At that point the energy is abruptly released, and a new cycle begins.

The description above fits the dynamic behavior of neuronal membranes. A richly structured empirical model of this behavior is found in the well-established differential formulation of Hodgkin and Huxley[6] and in a simplified version given by Fitzhugh[7]. These differential equations account for the foundations of neuronal activity and are also capable of representing subthreshold behavior and the refractoriness that follows each firing. When the membrane potential enters the critical region, an abrupt depolarization, i.e., a collapse of the potential difference across the membrane occurs followed by a somewhat slower recovery. This brief electrical

shorting of the membrane is called the action potential or "spike" and constitutes the output event for the neuron. If the causes for the initial depolarization are maintained, oscillation ( "limit-cycles") develops, generating multiple firings. Depending on input level and membrane parameters, the oscillation can be limited to a single spike, or may produce an oscillatory burst, or even continually sustained activity.

The present model shares the same general properties but uses the much simpler description of relaxation oscillator illustrated on Figure 1.

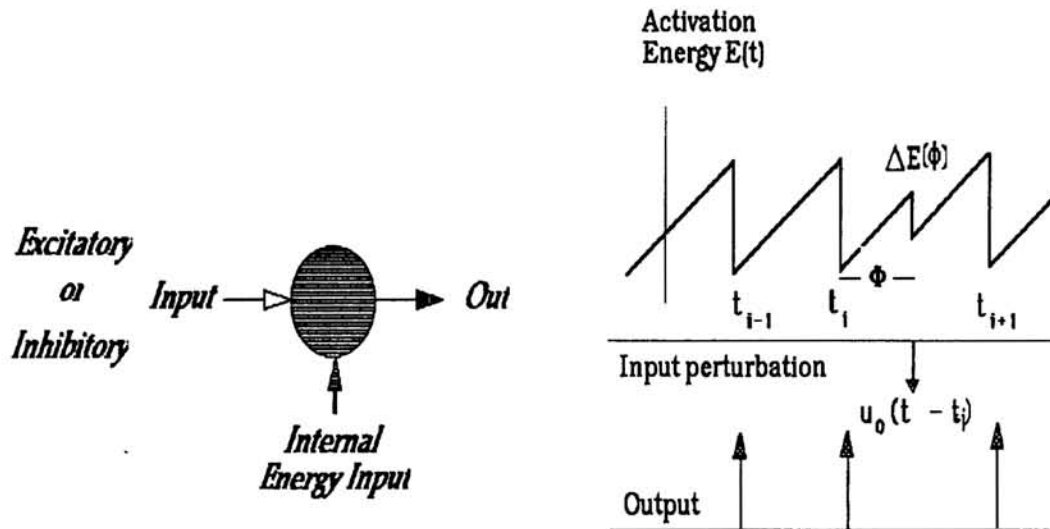

Figure 1 *Relaxation Oscillator with perturbation input*

Firing occurs when the energy level E(t) reaches some critical level $E_c$. Assuming a constant rate of energy influx **a**, firing will occur with the natural period

$$T = \frac{E_c.}{a.}$$

When pre-synaptic pulses impinge on the course of energy accumulation, the firing schedule is disturbed. Letting $t_0$ represent the instant of the last firing of the cell and tj, (j = 1,2,...J), the intants of impinging arrivals from other cells:

$$E(t - t_0) = a(t - t_0) + \Sigma w_j..u_0(t - t_i) \; ; \; E \le E_c$$

where $u_0(t)$ represents the unit impulse at **t=0**.

The dramatic complexity of synchronization dynamics can be appreciated by considering the simplest possible case, that of a master slave interaction between two regularly firing oscillator units A and B, with natural periods $T_A$ and $T_B$. At the instants of firing, unit A unidirectionally sends a spike signal to unit B which is received at some interval $\Phi$ measured from the last time B fired.

Upon reception the spike is transformed into a quantum of energy $\Delta E$ which depends upon the post-firing arrival time $\Phi$. The relationship $\Delta E(\Phi)$ can be shaped to represent refractoriness and other post-spike properties. Here it is assumed to be a simple ramp function. If the interaction is inhibitory, the consequence of this arrival is that the next firing of unit B is delayed (with respect to what its schedule would have been in absence of perturbation) by some positive interval $\delta$ (Figure 2). Because of the shape of $\Delta E(\Phi)$, the delaying action, nil immediately after firing, becomes longer for impinging pre-synaptic spikes that arrive later in the interval. If the interaction is excitatory, the delay is negative, i.e. a shortening of the natural firing interval. Under very general assumptions regarding the function $\Delta E(\Phi)$, B will tend to synchronize to A. Within a given range of coupling gains, the phase $\Phi$ will self-adjust until equilibrium is achieved. With a given $\Delta E(\Phi)$, this equilibrium corresponds to a distribution of maximum entropy, i.e., to the point where both cells receive the same amouint of activation, during their common cycle.

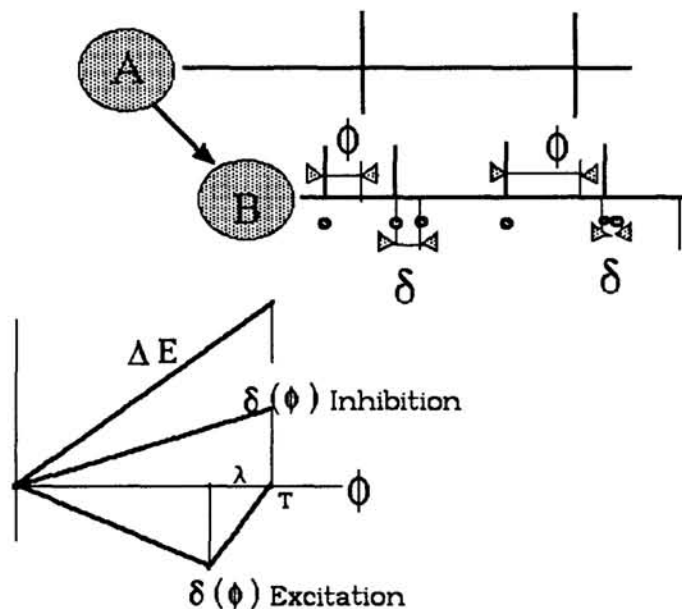

Figure 2 *Relationship between phase and delay when input efficiently increases linearly in the after-spike interval*

The synchronization dynamics presents an attractor for each rational frequency pair. To each ratio is associated a range of stability but only the ratios of lowest cardinality have wide zones of phase-locking (Figure 3). The wider stability zones correspond to a one to one ratio between $f_A$ and $f_B$ (or between their inverses $T_A$ and $T_B$). Kohn and Segundo have demonstrated that such phase locking occurs in living invertebrate neurons and pointed out the paradoxical nature of phase-locked inhibition which, within each stability region,

takes the appearance of excitation since small increases in input firing rate will locally result in increased output rates [8, 5].

The areas between these ranges of stability have the appearance of unstable transitions but in fact, as recently pointed out by Bak[9], form an infinity of locking steps known as the Devil's Staircase, corresponding to the infinity of intermediate rational pairs (figure 3). Bak showed that the staircase is self-similar under scaling and that the transitions form a fractal Cantor set with a fractal dimension which is a universal constant of dynamic systems.

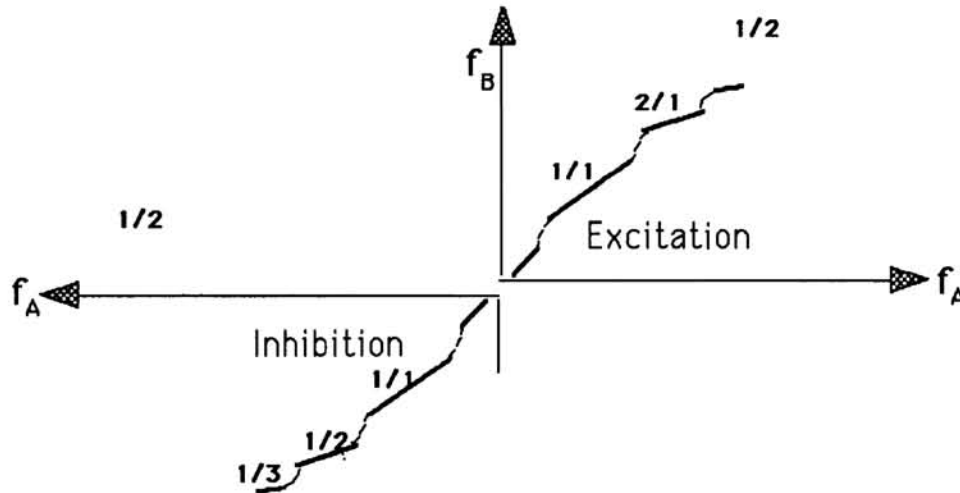

Figure 3 *Unilateral Synchronization:*

## CONSTRAINT SATISFACTION IN OSCILLATOR NETWORKS

The global synchronization of an interconnected network of mutually phase-locking oscillators is a constraint satisfaction problem. For each synchronization equilibrium, the nodes fire in interlocked patterns that organize inter-spike intervals into integer ratios.

The often cited "Traveling Salesman Problem", the archetype for a class of important "hard" problems, is a special case when the ratio must be 1/1; all nodes must fire at the same frequency. Here the equilibrium condition is that every node will accumulate the the same amount of energy during the global cycle. Furthermore, the firings must be ordered along a minimal path.

Using stochastic energy minimization and simulated annealing, the first simulations have demonstrated the feasibility of the approach with a limited number of nodes. The TSP is isomorphic to many other sequencing problems which involve distributed constraints.and fall into the oscillator array neural net paradigm in a particularly natural way. Work is being pursued to more rigorously establish the limits of applicability of the model..

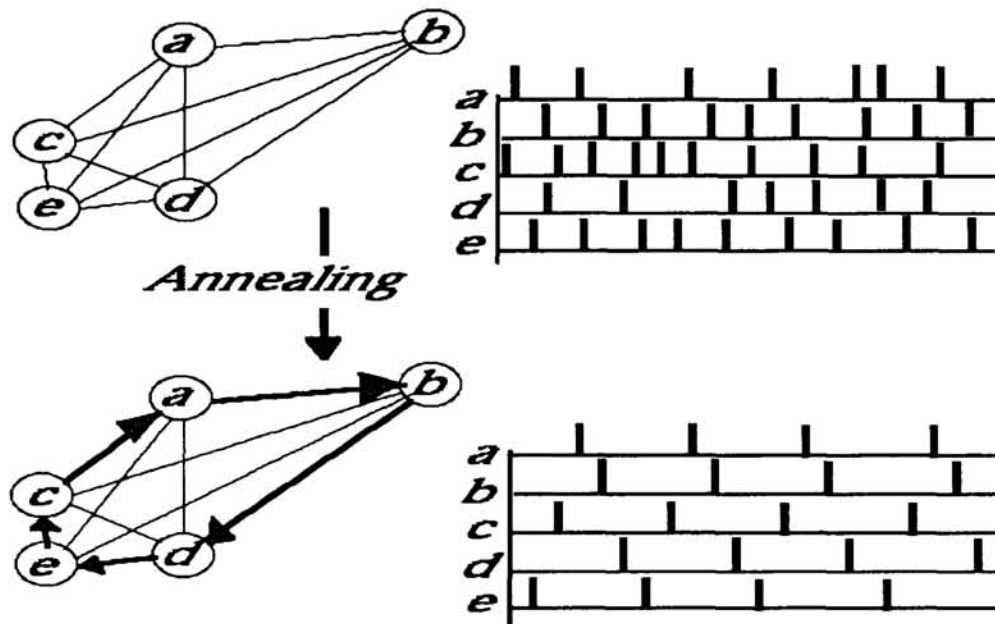

Figure 4. *The Traveling Salesman Problem: In the global oscillation of minimal energy each node is constrained to fire at the same rate in the order corresponding to the minimal path.*

## ACKNOWLEDGEMENT

Research supported in part by Aerojet Electro-Systems under the Aerojet-UCLA Cooperative Research Master Agreement No. D841211, and by NASA NAG 2-302.

## REFERENCES

1.  K. Fukushima, Biol. Cybern. **20**, 121 (1975).

2.  J.J. Hopfield, Proc. Nat. Acad. Sci. **79**, 2556 (1982).

3.  D.E. Rumelhart, G.E. Hinton, and R.J. Williams, *Parallel Distributed Processing: Explorations in the Microstructure of Cognition*, (MIT Press, Cambridge, MA., 1986) p. 318.

4.  J.P. Segundo, G.P. Moore, N.J. Stensaas, and T.H. Bullock, J. Exp. Biol. **40**, 643, (1963).

5.  J.P. Segundo and A.F. Kohn, Biol Cyber **40**, 113 (1981).

6.  A.L. Hodgkin and A.F. Huxley, J. Physiol. **117**, 500 (1952).

7.  Fitzhugh, Biophysics J., **1**, 445 (1961).

8.  A.F. Kohn, A. Freitas da Rocha, and J.P. Segundo, Biol. Cybern. **41**, 5 (1981).

9.  P. Bak, Phys. Today (Dec 1986) p. 38 .

10. J. Haggerty and J.J. Vidal, UCLA BCI Report, 1975.
